# Learning Mixture Hierarchies

**Nuno Vasconcelos**        **Andrew Lippman**
MIT Media Laboratory, 20 Ames St, E15-320M, Cambridge, MA 02139,
{nuno,lip}@media.mit.edu,        http://www.media.mit.edu/~nuno

## Abstract

The hierarchical representation of data has various applications in domains such as data mining, machine vision, or information retrieval. In this paper we introduce an extension of the *Expectation-Maximization* (EM) algorithm that learns mixture hierarchies in a computationally efficient manner. Efficiency is achieved by progressing in a bottom-up fashion, i.e. by clustering the mixture components of a given level in the hierarchy to obtain those of the level above. This clustering requires only knowledge of the mixture parameters, there being no need to resort to intermediate samples. In addition to practical applications, the algorithm allows a new interpretation of EM that makes clear the relationship with non-parametric *kernel-based* estimation methods, provides explicit control over the trade-off between the bias and variance of EM estimates, and offers new insights about the behavior of deterministic annealing methods commonly used with EM to escape local minima of the likelihood.

## 1  Introduction

There are many practical applications of statistical learning where it is useful to characterize data hierarchically. Such characterization can be done according to either top-down or bottom-up strategies. While the former start by generating a coarse model that roughly describes the entire space, and then successively refine the description by partitioning the space and generating sub-models for each of the regions in the partition; the later start from a fine description, and successively agglomerate sub-models to generate the coarser descriptions at the higher levels in the hierarchy.

Bottom-up strategies are particularly useful when not all the data is available at once, or when the dataset is so big that processing it as whole is computationally infeasible. This is the case of machine vision tasks such as object recognition, or the indexing of video databases. In object recognition, it is many times convenient to determine not only which object is present in the scene but also its pose [2], a goal that can be attained by a hierarchical, description where at the lowest level a model is learned for each object pose and all pose models are then combined into a generic model at the top level of the hierarchy. Similarly,

for video indexing, one may be interested in learning a description for each frame and then combine these into shot descriptions or descriptions for some other sort of high level temporal unit [6].

In this paper we present an extension of the EM algorithm [1] for the estimation of hierarchical mixture models in a bottom-up fashion. It turns out that the attainment of this goal has far more reaching consequences than the practical applications above. In particular, because a kernel density estimate can be seen as a limiting case of a mixture model (where a mixture component is superimposed on each sample), this extension establishes a direct connection between so-called parametric and non-parametric density estimation methods making it possible to exploit results from the vast non-parametric smoothing literature [4] to improve the accuracy of parametric estimates. Furthermore, the original EM algorithm becomes a particular case of the one now presented, and a new intuitive interpretation becomes available for an important variation of EM (known as *deterministic annealing*) that had previously been derived from statistical physics. With regards to practical applications, the algorithm leads to computationally efficient methods for estimating density hierarchies capable of describing data at different resolutions.

## 2 Hierarchical mixture density estimation

Our model consists of a hierarchy of mixture densities, where the data at a given level is described by

$$P(\mathbf{X}) = \sum_{k=1}^{C^l} \pi_k^l p(\mathbf{X}|z_k^l = 1, \mathcal{M}_l), \tag{1}$$

where $l$ is the level in the hierarchy ($l = 0$ providing the coarsest characterization of the data), $\mathcal{M}_l$ the mixture model at this level, $C^l$ the number of mixture components that compose it, $\pi_k^l$ the prior probability of the $k^{th}$ component, and $z_k^l$ a binary variable that takes the value 1 if and only if the sample $\mathbf{X}$ was drawn from this component. The only restriction on the model is that if node $j$ of level $l + 1$ is a child of node $i$ of level $l$, then

$$\pi_j^{l+1} = \pi_{j|k}^{l+1} \pi_k^l, \tag{2}$$

where $k$ is the parent of $j$ in the hierarchy of hidden variables.

The basic problem is to compute the mixture parameters of the description at level $l$ given the knowledge of the parameters at level $l + 1$. This can also be seen as a problem of clustering mixture components. A straightforward solution would be to draw a sample from the mixture density at level $l + 1$ and simply run EM with the number of classes of the level $l$ to estimate the corresponding parameters. Such a solution would have at least two major limitations. First, there would be no guarantee that the constraint of equation (2) would be enforced, i.e. there would be no guarantee of structure in the resulting mixture hierarchy, and second it would be computationally expensive, as all the models in the hierarchy would have to be learned from a large sample. In the next section, we show that this is really not necessary.

## 3 Estimating mixture hierarchies

The basic idea behind our approach is, instead of generating a real sample from the mixture model at level $l + 1$, to consider a *virtual sample* generated from the same model, use EM to find the expressions for the parameters of the mixture model of level $l$ that best explain this virtual sample, and establish a closed-form relationship between these parameters and those of the model at level $l + 1$. For this, we start by considering a virtual sample $\mathbf{X} = \{\mathbf{X}_1, \ldots, \mathbf{X}_{C^{l+1}}\}$ from $\mathcal{M}_{l+1}$, where each of the $\mathbf{X}_i$ is a virtual sample from one of

the $C^{l+1}$ components of this model, with size $M_i = \pi_i^l N$, where $N$ is the total number of virtual points.

We next establish the likelihood for the virtual sample under the model $\mathcal{M}_l$. For this, as is usual in the EM literature, we assume that samples from different blocks are independent, i.e.

$$P(\mathbf{X}|\mathcal{M}_l) = \prod_{i=1}^{C^{l+1}} P(\mathbf{X}_i|\mathcal{M}_l), \tag{3}$$

but, to ensure that the constraint of equation (2) is enforced, samples within the same block are assigned to the same component of $\mathcal{M}_l$. Assuming further that, given the knowledge of the assignment the samples are drawn independently from the corresponding mixture component, the likelihood of each block is given by

$$P(\mathbf{X}_i|\mathcal{M}_l) = \sum_{j=1}^{C^l} \pi_j^l P(\mathbf{X}_i|z_{ij}=1,\mathcal{M}_l) = \sum_{j=1}^{C^l} \pi_j^l \prod_{m=1}^{M_i} P(\mathbf{x}_i^m|z_{ij}=1,\mathcal{M}_l), \tag{4}$$

where $z_{ij} = z_i^{l+1} z_j^l$ is a binary variable with value one if and only if the block $\mathbf{X}_i$ is assigned to the $j^{th}$ component of $\mathcal{M}_l$, and $\mathbf{x}_i^m$ is the $m^{th}$ data point in $\mathbf{X}_i$. Combining equations (3) and (4) we obtain the *incomplete data* likelihood, under $\mathcal{M}_l$, for the whole sample

$$P(\mathbf{X}|\mathcal{M}_l) = \prod_{i=1}^{C^{l+1}} \sum_{j=1}^{C^l} \pi_j^l \prod_{m=1}^{M_i} P(\mathbf{x}_i^m|z_{ij}=1,\mathcal{M}_l). \tag{5}$$

This equation is similar to the incomplete data likelihood of standard EM, the main difference being that instead of having an hidden variable for each sample point, we now have one for each sample block. The likelihood of the *complete data* is given by

$$P(\mathbf{X},\mathbf{Z}|\mathcal{M}_l) = \prod_{i=1}^{C^{l+1}} \prod_{j=1}^{C^l} \left[\pi_j^l P(\mathbf{X}_i|z_{ij}=1,\mathcal{M}_l)\right]^{z_{ij}}, \tag{6}$$

where $\mathbf{Z}$ is a vector containing all the $z_{ij}$, and the log-likelihood becomes

$$\log P(\mathbf{X},\mathbf{Z}|\mathcal{M}_l) = \sum_{i=1}^{C^{l+1}} \sum_{j=1}^{C^l} z_{ij} \log(\pi_j^l P(\mathbf{X}_i|z_{ij}=1,\mathcal{M}_l)). \tag{7}$$

Relying on EM to estimate the parameters of $\mathcal{M}_l$ leads to the the following E-step

$$h_{ij} = E[z_{ij}|\mathbf{X}_i,\mathcal{M}_l] = P(z_{ij}=1|\mathbf{X}_i,\mathcal{M}_l) = \frac{P(\mathbf{X}_i|z_{ij}=1,\mathcal{M}_l)\pi_j^l}{\sum_k P(\mathbf{X}_i|z_{ik}=1,\mathcal{M}_l)\pi_k^l}. \tag{8}$$

The key quantity to compute is therefore $P(\mathbf{X}_i|z_{ij}=1,\mathcal{M}_l)$. Taking its logarithm

$$\log P(\mathbf{X}_i|z_{ij}=1,\mathcal{M}_l) = M_i[\frac{1}{M_i}\sum_{i=1}^{M_i} \log P(\mathbf{x}_i^m|z_{ij}=1,\mathcal{M}_l)]$$

$$= M_i E_{\mathcal{M}_{l+1,i}}[\log P(\mathbf{x}|z_{ij}=1,\mathcal{M}_l)], \tag{9}$$

where we have used the law of large numbers, and $E_{\mathcal{M}_{l+1,i}}[\mathbf{x}]$ is the expected value of $\mathbf{x}$ according the $i^{th}$ mixture component of $\mathcal{M}_{l+1}$ (the one from which $\mathbf{X}_i$ was drawn). This is an easy computation for most densities commonly used in mixture modeling. It can be shown [5] that for the Gaussian case it leads to

$$h_{ij} = \frac{\left[\mathcal{G}(\mu_i^{l+1},\mu_j^l,\Sigma_j^l)e^{-\frac{1}{2}trace\{(\Sigma_j^l)^{-1}\Sigma_i^{l+1}\}}\right]^{M_i} \pi_j^l}{\sum_k \left[\mathcal{G}(\mu_i^{l+1},\mu_k^l,\Sigma_k^l)e^{-\frac{1}{2}trace\{(\Sigma_k^l)^{-1}\Sigma_i^{l+1}\}}\right]^{M_i} \pi_k^l}, \tag{10}$$

where $\mathcal{G}(\mathbf{x}, \mu, \Sigma)$ is the expression for a Gaussian with mean $\mu$ and covariance $\Sigma$.

The M-step consists of maximizing

$$Q = \sum_{i=1}^{C^{l+1}} \sum_{j=1}^{C^l} h_{ij} \log(\pi_j^l P(\mathbf{X}_i | z_{ij} = 1, \mathcal{M}_l)) \tag{11}$$

subject to the constraint $\sum_j \pi_j^l = 1$. Once again, this is a relatively simple task for common mixture models and in [5] we show that for the Gaussian case it leads to the following parameter update equations

$$\pi_j^l = \frac{\sum_i h_{ij}}{C^{l+1}} \tag{12}$$

$$\mu_j^l = \frac{\sum_i h_{ij} M_i \mu_i^{l+1}}{\sum_i h_{ij} M_i} \tag{13}$$

$$\Sigma_j^l = \frac{1}{\sum_i h_{ij} M_i} \left[ \sum_i h_{ij} M_i \Sigma_i^{l+1} + \sum_i h_{ij} M_i (\mu_i^{l+1} - \mu_j^l)(\mu_i^{l+1} - \mu_j^l)^T \right] . \tag{14}$$

Notice that neither equation (10) nor equations (12) to (14) depend explicitly on the underlying sample $\mathbf{X}_i$ and can be computed directly from the parameters of $\mathcal{M}_{l+1}$. The algorithm is thus very efficient from a computational standpoint as the number of mixture components in $\mathcal{M}_{l+1}$ is typically much smaller than the size of the sample at the bottom of the hierarchy.

## 4  Relationships with standard EM

There are interesting relationships between the algorithm derived above and the standard EM procedure. The first thing to notice is that by making $M_i = 1$ and $\Sigma_i^{l+1} = 0$, the E and M-steps become those obtained by applying standard EM to the sample composed of the points $\mu_i^{l+1}$.

Thus, standard EM can be seen as a particular case of the new algorithm, that learns a two level mixture hierarchy. An initial estimate is first obtained at the bottom of this hierarchy by placing a Gaussian with zero covariance on top of each data point, the model at the second level being then computed from this estimate. The fact that the estimate at the bottom level is nothing more than a kernel estimate with zero bandwidth suggests that other choices of the kernel bandwidth may lead to better overall EM estimates.

Under this interpretation, the $\Sigma_i^{l+1}$ become free parameters that can be used to control the smoothness of the density estimates and the whole procedure is equivalent to the composition of three steps: 1) find the kernel density estimate that best fits the sample under analysis, 2) draw a larger virtual sample from that density, and 3) compute EM estimates from this larger sample. In section 5, we show that this can leave to significant improvements in estimation accuracy, particularly when the initial sample is small, the free parameters allowing explicit control over the trade-off between the bias and variance of the estimator.

Another interesting relationship between the hierarchical method and standard EM can be derived by investigating the role of the size of the underlying virtual sample (which determines $M_i$) on the estimates. Assuming $M_i$ constant, $M_i = M, \forall i$, it factors out of all summations in equations (12) to (14), the contributions of numerator and denominator canceling each other. In this case, the only significance of the choice of $M$ is its impact on the E-step. Assuming, as before, that $\Sigma_i^{l+1} = 0$ we once again have the EM algorithm, but where the class-conditional likelihoods of the E-step are now raised to the $M^{th}$ power. If

$M$ is seen as the inverse of temperature, both the E and M steps become those of standard EM under deterministic annealing (DA) [1] [3].

The DA process is therefore naturally derived from our hierarchical formulation, which gives it a new interpretation that is significantly simpler and more intuitive than those derived from statistical physics. At the start of the process $M$ is set to zero, i.e. no virtual samples are drawn from the Gaussian superimposed on the real dataset, and there is no virtual data. Thus, the assignments $h_{ij}$ of the E-step simply become the prior mixing proportions $\pi_j^l$ and the M-step simply sets the parameters of all Gaussians in the model to the sample mean and sample covariance of the real sample. As $M$ increases, the number of virtual points drawn from each Gaussian also increases and for $M = 1$ we have a single point that coincides with the point on the real training sample. We therefore obtain the standard EM equations. Increasing $M$ further will make the E-step assignments harder (in the limit of $M = \infty$ each point is assigned to a single mixture component) because a larger virtual probability mass is attached to each real point leading to much higher certainty with regards to the reliability of the assignment.

Overall, while in the beginning of the process the reduced size of the virtual sample allows the points in the real sample to switch from mixture to mixture easily, as $M$ is increased the switching becomes much less likely. The "exploratory" nature of the initial iterations drives the process towards solutions that are globally good, therefore allowing it to escape local minima.

## 5   Experimental results

In this section, we present experimental results that illustrate the properties of the hierarchical EM algorithm now proposed. We start by a simple example that illustrates how the algorithm can be used to estimate hierarchical mixtures.

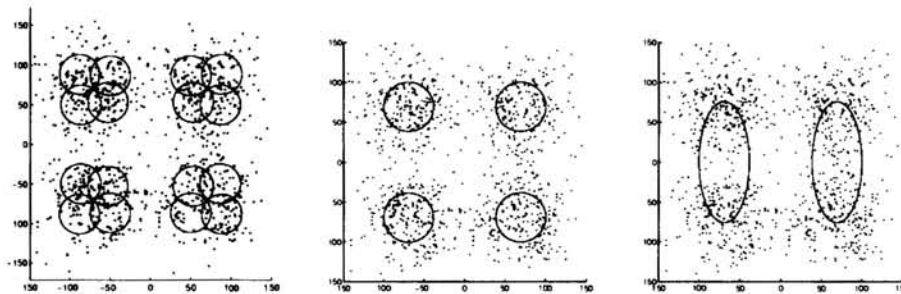

Figure 1: Mixture hierarchy derived from the model shown in the left. The plot relative to each level of the hierarchy is superimposed on a sample drawn from this model. Only the one-standard deviation contours are shown for each Gaussian.

The plot on the left of Figure 1 presents a Gaussian mixture with 16 uniformly weighted components. A sample with 1000 points was drawn from this model, and the algorithm used to find the best descriptions for it at three resolutions (mixtures with 16, 4, and 2 Gaussian). These descriptions are shown in the figure. Notice how the mixture hierarchy naturally captures the various levels of structure exhibited by the data.

This example suggests how the algorithm could be useful for applications such as object recognition or image retrieval. Suppose that each of the Gaussians in the leftmost plot of

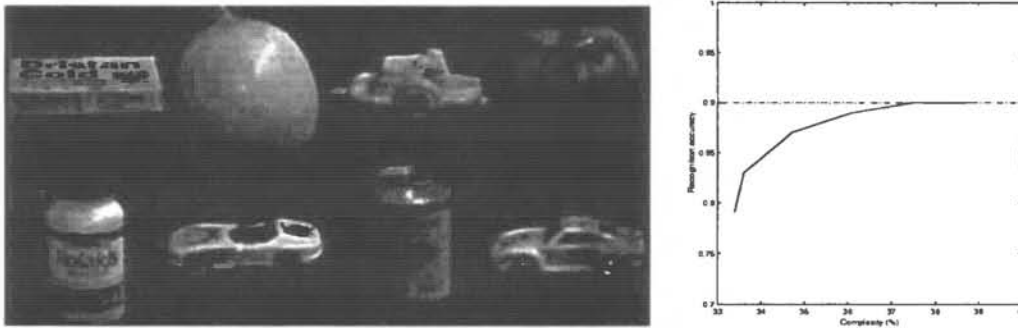

Figure 2: Object recognition task. Left: 8 of the 100 objects in the database. Right: computational savings achieved with hierarchical recognition vs full search.

the figure describes how a given pose of a given object populates a 2-D feature space on which object recognition is to be performed. In this case, higher levels in the hierarchical representation provide a more generic description of the object. E.g. each of the Gaussians in the model shown in the middle of the figure might provide a description for all the poses in which the camera is on the same quadrant of the viewing sphere, while those in the model shown in the right might represent views from the same hemisphere. The advantage, for recognition or retrieval, of relying on a hierarchal structure is that the search can be performed first at the highest resolution, where it is much less expensive, only the best matches being considered at the subsequent levels.

Figure 2 illustrates the application of hierarchical mixture modeling to a real object recognition task. Shown on the left side of the figure are 8 objects from the 100 contained in the Columbia object database [2]. The database consists of 72 views (obtained by positioning the camera in $5^o$ intervals along a circle on the viewing sphere), which were evenly separated into a training and a test set. A set of features was computed for each image, and a hierarchical model was then learned for each object in the resulting feature space. While the process could be extended to any number of levels, here we only report on the case of a two-level hierarchy: at the bottom each image is described by a mixture of 8 Gaussians, and at the top each mixture (also with 8 Gaussians) describes 3 consecutive views. Thus, the entire training set is described by 3600 mixtures at the bottom resolution and 1200 at the top.

Given an image of an object to recognize, recognition takes place by computing its projection into the feature space, measuring the likelihood of the resulting sample according to each of the models in the database, and choosing the most likely. The complexity of the process is proportional to the database size. The plot on the left of Figure 2 presents the recognition accuracy achieved with the hierarchical representation vs the corresponding complexity, shown as a percent of the complexity required by full search. The full-search accuracy is in this case 90%, and is also shown as a straight line in the graph. As can be seen from the figure, the hierarchical search achieves the full search accuracy with less than 40% of its complexity. We are now repeating this experiments with deeper trees, where we expect the gains to be even more impressive.

We finalize by reporting on the impact of smoothing on the quality of EM estimates. For this, we conducted the following Monte Carlo experiment: 1) draw 200 datasets $\mathcal{S}_i, i = 1, \ldots, 200$ from the model shown on the left of Figure 1, 2) fit each dataset with EM, 3) measure the correlation coefficient $\rho_i, i = 1, \ldots, 200$ between each of the EM fits and the original model, and 4) compute the sample mean $\hat{\rho}$ and variance $\hat{\sigma}_\rho$. The correlation coefficient is defined by $\rho_i = \int f(\mathbf{x})\hat{f}_i(\mathbf{x})d\mathbf{x}/(\int f(\mathbf{x})d\mathbf{x} \int \hat{f}_i(\mathbf{x})d\mathbf{x})$, where $f(\mathbf{x})$ is the

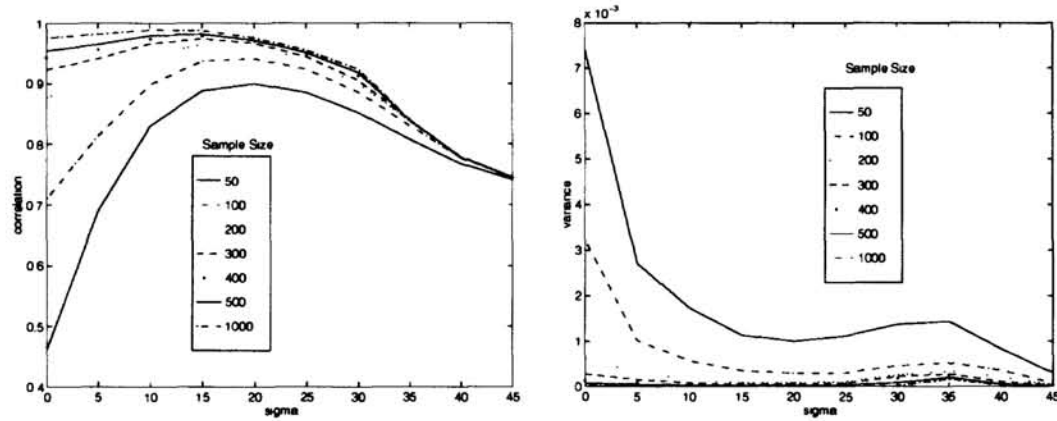

Figure 3: Results of the Monte Carlo experiment described on the text. Left: $\hat{\rho}$ as a function $\sigma_k$. Right: $\hat{\sigma}_\rho$ as a function of $\sigma_k$. The various curves in each graph correspond to to different sample sizes.

true model and $f_i(\mathbf{x})$ the $i^{th}$ estimate, and can be computed in closed form for Gaussian mixtures. The experiment was repeated with various dataset sizes and various degrees of smoothing (by setting the bandwidth of the underlying Gaussian kernel to $\sigma_k^2 \mathbf{I}$ for various values of $\sigma_k$).

Figure 3 presents the results of this experiment. It is clear, from the graph on the left, that smoothing can have a significant impact on the quality of the EM estimates. This impact is largest for small samples, where smoothing can provide up to a two fold improvement estimation accuracy, but can be found even for large samples.

The kernel bandwidth allows control over the trade-off between the bias and variance of the estimates. When $\sigma_k$ is zero (standard EM), bias is small but variance can be large, as illustrated by the graph on the right of the figure. As $\sigma_k$ is increased, variance decreases at the cost of an increase in bias (the reason why for large $\sigma_k$ all lines in the graph of the left meet at the same point regardless of the sample size). The point where $\hat{\rho}$ is the highest is the point at which the bias-variance trade off is optimal. Operating at this point leads to a much smaller dependence of the accuracy of the estimates on the sample size or, conversely, the need for much smaller samples to achieve a given degree of accuracy.

## Footnotes

[1] DA is a technique drawn from analogies with statistical physics that avoids local maxima of the likelihood function (in which standard EM can get trapped) by performing a succession of optimizations at various temperatures [3].

## References

[1] A. Dempster, N. Laird, and D. Rubin. Maximum-likelihood from Incomplete Data via the EM Algorithm. *J. of the Royal Statistical Society*, B-39, 1977.

[2] H. Murase and S. Nayar. Visual Learning and Recognition of 3-D Objects from Appearence. *International Journal of Computer Vision*, 14:5–24, 1995.

[3] K. Rose, E. Gurewitz, and G. Fox. Vector Quantization by Determinisc Annealing. *IEEE Trans. on Information Theory*, Vol. 38, July 1992.

[4] J. Simonoff. *Smoothing Methods in Statistics*. Springer-Verlag, 1996.

[5] N. Vasconcelos and A. Lippman. Learning Mixture Hierarchies. Technical report, MIT Media Laboratory, 1998. Available from ftp://ftp.media.mit.edu/pub/nuno/HierMix.ps.gz.

[6] N. Vasconcelos and A. Lippman. Content-based Pre-Indexed Video. In *Proc. Int. Conf. Image Processing*, Santa Barbara, California, 1997.
